# ACh, Uncertainty, and Cortical Inference

**Peter Dayan**     **Angela Yu**
Gatsby Computational Neuroscience Unit
17 Queen Square, London, England, WC1N 3AR.
dayan@gatsby.ucl.ac.uk     feraina@gatsby.ucl.ac.uk

## Abstract

Acetylcholine (ACh) has been implicated in a wide variety of tasks involving attentional processes and plasticity. Following extensive animal studies, it has previously been suggested that ACh reports on *uncertainty* and controls hippocampal, cortical and cortico-amygdalar plasticity. We extend this view and consider its effects on cortical representational inference, arguing that ACh controls the balance between bottom-up inference, influenced by input stimuli, and top-down inference, influenced by contextual information. We illustrate our proposal using a hierarchical hidden Markov model.

## 1   Introduction

The individual and joint computational roles of neuromodulators such as dopamine, serotonin, norepinephrine and acetylcholine are currently the focus of intensive study.[5,7,9–11,16,27] A rich understanding of the effects of neuromodulators on the dynamics of networks has come about through work in invertebrate systems.[21] Further, some general computational ideas have been advanced, such as that they change the signal to noise ratios of cells. However, more recent studies, particularly those focusing on dopamine,[26] have concentrated on specific computational tasks.

ACh was one of the first neuromodulators to be attributed a specific role. Hasselmo and colleagues,[10,11] in their seminal work, proposed that cholinergic (and, in their later work, also GABAergic[12]) modulation controls read-in to and read-out from recurrent, attractor-like memories, such as area CA3 of the hippocampus. Such memories fail in a characteristic manner if the recurrent connections are operational during storage, thus forcing new input patterns to be mapped to existing memories. Not only would these new patterns lose their specific identity, but, worse, through standard synaptic plasticity, the size of the basin of attraction of the offending memory would actually be increased, making similar problems *more* likely. Hasselmo *et al* thus suggested, and collected theoretical and experimental evidence in favor of, the notion that ACh (from the septum) should control the suppression and plasticity of specific sets of inputs to CA3 neurons. During read-in, high levels of ACh would suppress the recurrent synapses, but make them readily plastic, so that new memories would be stored without being pattern-completed. Then, during read-out, low levels of ACh would boost the impact of the recurrent weights (and reduce their plasticity), allowing auto-association to occur.

The ACh signal to the hippocampus can be characterized as reporting the *unfamiliarity* of the input with which its release is associated. This is analogous to its

characterization as reporting the *uncertainty* associated with predictions in theories of attentional influences over learning in classical conditioning.[4] In an extensive series of investigations in rats, Holland and his colleagues[14,15] have shown that a cholinergic projection from the nucleus basalis to the (parietal) cortex is important when animals have to devote more learning (which, in conditioning, is essentially synonymous with paying incremental attention) to stimuli about whose consequences the animal is uncertain.[20] We have[4] interpreted this in the statistical terms of a Kalman filter, arguing that the ACh signal reported this uncertainty, thus changing plasticity appropriately. Note, however, that unlike the case of the hippocampus, the mechanism of action of ACh in conditioning is not well understood.

In this paper, we take the idea that ACh reports on uncertainty one step farther. There is a wealth of analysis-by-synthesis unsupervised learning models of cortical processing.[1,3,8,13,17,19,23] In these, top-down connections instantiate a *generative* model of sensory input; and bottom-up connections instantiate a *recognition* model, which is the statistical inverse of the generative model, and maps inputs into categories established in the generative model. These models, at least in principle, permit stimuli to be processed according both to bottom-up input and top-down expectations, the latter being formed based on temporal context or information from other modalities. Top-down expectations can resolve bottom-up ambiguities, permitting better processing. However, in the face of contextual *uncertainty*, top-down information is useless. We propose that ACh reports on top-down uncertainty, and, as in the case of area CA3, differentially modulates the strength of synaptic connections: comparatively weakening those associated with the top-down generative model, and enhancing those associated with bottom-up, stimulus-bound information.[2] Note that this interpretation is broadly consistent with existent electrophysiology data, and documented effects on stimulus processing of drugs that either enhance (*eg* cholinesterase inhibitors) or suppress (*eg* scopolamine) the action of ACh.[6,25,28]

There is one further wrinkle. In exact bottom-up, top-down, inference using a generative model, top-down contextual uncertainty does not play a simple role. Rather, all possible contexts are treated simultaneously according to the individual posterior probabilities that they currently pertain. Given the neurobiologically likely scenario in which one set of units has to be used to represent all possible contexts, this exact inferential solution is not possible. Rather, we propose that a single context is represented in the activities of high level (presumably pre-frontal) cortical units, and uncertainty associated with this context is represented by ACh. This cholinergic signal then controls the balance between bottom-up and top-down influences over inference.

In the next section, we describe the simple hierarchical generative model that we use to illustrate our proposal. The ACh-based recognition model is introduced in section 3 and discussed in section 4.

## 2   Generative and Recognition Models

Figure 1A shows a very simple case of a hierarchical generative model. The generative model is a form of hidden Markov model (HMM), with a discrete hidden state $z_t$, which will capture the idea of a persistent temporal context, and a two-dimensional, real-valued, output $\mathbf{x}_t$. Crucially, there is an extra $y$ layer, between $z$ and $\mathbf{x}$. The state $y_t$ is stochastically determined from $z_t$, and controls which of a set of 2d Gaussians (centered at the corners of the unit square) is used to generate $\mathbf{x}_t$. In this austere case, $y_t$ is the model's *representation* of $\mathbf{x}_t$, and the key inference

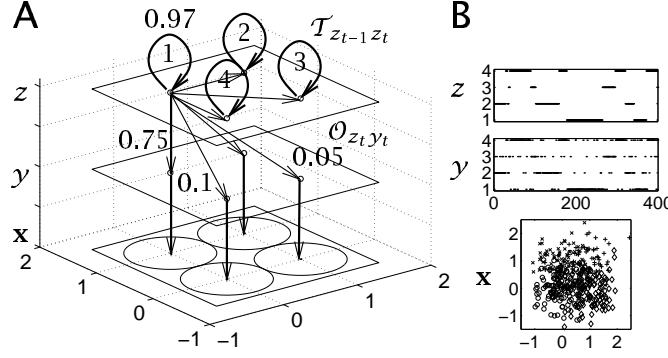

**Figure 1**: Generative model. A) Three-layer model $z \in \{1-4\} \Rightarrow y \in \{1-4\} \Rightarrow \mathbf{x} \in \Re^2$ with dynamics ($\mathcal{T}$) in the $z$ layer ($P[z_t = z_{t-1}] = 0.97$), a probabilistic mapping ($\mathcal{O}$) from $z \to y$ ($P[y_t = z_t|z_t] = 0.75$), and a Gaussian model $p[\mathbf{x}|y]$ with means at the corners of the unit square and standard deviation $0.5$ in each direction. The model is rotationally invariant; only some of the links are shown for convenience. B) Sample sequence showing the slow dynamics in $z$; the stochastic mapping into $y$ and the substantial overlap in $\mathbf{x}$ (different symbols show samples from the different Gaussians shown in A).

problem will be to determine the distribution over which $y_t$ generated $\mathbf{x}_t$, given the past experience $\mathcal{D}_{t-1} = \{\mathbf{x}_1, \dots, \mathbf{x}_{t-1}\}$ and $\mathbf{x}_t$ itself.

Figure 1B shows an example of a sequence of 400 steps generated from the model. The state in the $z$ layer stays the same for an average of about 30 timesteps; and then switches to one of the other states, chosen equally at random. The transition matrix is $\mathcal{T}_{z_{t-1}z_t}$. The state in the $y$ layer is more weakly determined by the state in the $z$ layer, with a probability of only $3/4$ that $y_t = z_t$. The stochastic transition from $z$ to $y$ is governed by the transition matrix $\mathcal{O}_{z_t y_t}$. Finally, $\mathbf{x}_t$ is generated as a Gaussian about a mean specified by $y_t$. The standard deviation of these Gaussians ($0.5$ in each direction) is sufficiently large that the densities overlap substantially.

The naive solution to inferring $y_t$ is to use only the likelihood term (*ie* only the probabilities $P[\mathbf{x}_t|y_t]$). The performance of this is likely to be poor, since the Gaussians in $\mathbf{x}$ for the different values of $y$ overlap so much. However, and this is why it is a paradigmatic case for our proposal, contextual information, in this case past experience, can help to determine $y_t$. We show how the putative effect of ACh in controlling the balance between bottom-up and top-down inference in this model can be used to build a good approximate inference model.

In order to evaluate our approximate model, we need to understand optimal inference in this case. Figure 2A shows the standard HMM inference model, which calculates the exact posteriors $P[y_t|\mathcal{D}_t]$ and $P[z_t|\mathcal{D}_t]$. This is equivalent to just the forward part of the forwards-backwards algorithm[22] (since we are not presently interested in learning the parameters of the model). The adaptation to include the $y$ layer is straightforward. Figures 3A;D;E show various aspects of exact inference for a particular run. The histograms in figure 3A show that $P[y_t|\mathcal{D}_t]$ captures quite well the actual states $y_t^*$ that generated the data. The upper plot shows the posterior probabilities of the actual states in the sequence – these should be, and are, usually high; the lower histogram the posterior probability of the other possible states; these should be, and are, usually low. Figure 3D shows the actual *state sequence* $z_t^*$; figure 3E shows the states that are individually most likely at each time step (note that this is not the maximum likelihood state sequence, as found by the Viterbi algorithm, for instance).

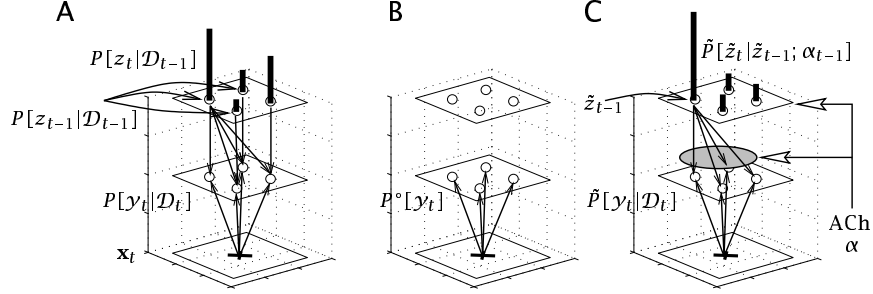

**Figure 2**: Recognition models. A) Exact recognition model. $P[z_{t-1}|\mathcal{D}_{t-1}]$ is propagated to provide the prior $P[z_t|\mathcal{D}_{t-1}]$ (shown by the lengths of the thick vertical bars) and thus the prior $P[y_t|\mathcal{D}_{t-1}]$. This is combined with the likelihood term from the data $\mathbf{x}_t$ to give the true $P[y_t|\mathcal{D}_t]$. B) Bottom-recognition model uses only a generic prior over $y_t$ (which conveys no information), and so the likelihood term dominates. C) ACh model. A single estimated state $\tilde{z}_{t-1}$ is used, in conjunction with its certainty $\alpha_{t-1}$, reported by cholinergic activity, to produce an approximate prior $\tilde{P}[\tilde{z}_t|\tilde{z}_{t-1}]$ over $z_t$ (which is a mixture of a delta function and a uniform), and thus an approximate prior over $y_t$. This is combined with the likelihood to give an approximate $\tilde{P}[y_t|\mathcal{D}_t]$, and a new cholinergic signal $\alpha_t$ is calculated.

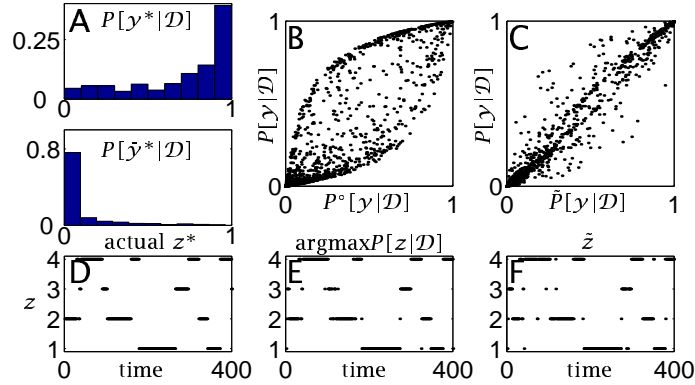

**Figure 3**: Exact and approximation recognition. A) Histograms of the exact posterior distribution $P[y|\mathcal{D}]$ over the actual state $y_t^*$ (upper) and the other possible states $y \neq y_t^*$ (lower, written $P[\bar{y}^*]$). This shows the quality of exact representational inference. B;C) Comparison of the purely bottom up $P^{\circ}[y_t|\mathbf{x}_t]$ (B) and the ACh-based approximation $\tilde{P}[y_t|\mathcal{D}]$ (C) with the true $P[y_t|\mathcal{D}]$ across all values of $y$. The ACh-based approximation is substantially more accurate. D) Actual $z_t$. E) Highest probability $z$ state from the exact posterior distribution. F) Single $\tilde{z}$ state in the ACh model.

Figure 2B shows a purely bottom up model that only uses the likelihood terms to infer the distribution over $y_t$. This has $P^{\circ}[y_t|\mathbf{x}_t] = p[\mathbf{x}_t|y_t]/\mathcal{Z}$ where $\mathcal{Z}$ is a normalization factor. Figure 3B shows the representational performance of this model, through a scatter-plot of $P^{\circ}[y_t|\mathbf{x}_t]$ against the exact posterior $P[y_t|\mathcal{D}_t]$. If bottom-up inference was correct, then all the points would lie on the line of equality – the bow-shape shows that purely bottom-up inference is relatively poor. Figure 4C shows this in a different way, indicating the difference between the average summed log probabilities of the actual states under the bottom up model and those under the true posterior. The larger and more negative the difference, the worse the approximate inference. Averaging over 1000 runs, the difference is $-70$ log units (compared with a total log likelihood under the exact model of $-210$).

## 3 ACh Inference Model

Figure 2C shows the ACh-based approximate inference model. The information about the context comes in the form of two quantities: $\tilde{z}_{t-1}$, the approximated contextual *state* having seen $\mathcal{D}_{t-1}$, and $\alpha_{t-1}$, which is the measure of *uncertainty* in that contextual state. The idea is that $\alpha_{t-1}$ is reported by ACh, and is used to control (indicated by the filled-in ellipse) the extent to which top-down information based on $\tilde{z}_{t-1}$ is used to influence inference about $y_t$. If we were given the full exact posterior distribution $P[z_{t-1}, y_{t-1}|\mathcal{D}_{t-1}]$, then one natural definition for this ACh signal would be the uncertainty in the most likely contextual state

$$\alpha_{t-1} = 1 - \max_z P[z_{t-1} = z|\mathcal{D}_{t-1}] \tag{1}$$

Figure 4A shows the resulting ACh signal for the case of figure 3. As expected, ACh is generally high at times when the true state $z_t^*$ is changing, and decreases during the periods that $z_t^*$ is constant. During times of change, top-down information is confusing or potentially incorrect, and so bottom-up information should dominate. This is just the putative inferential effect of ACh.

However, the ACh signal of figure 4A was calculated assuming knowledge of the true posterior, which is unreasonable. The model of figure 2C includes the key approximation that the only other information from $\mathcal{D}_{t-1}$ about the state of $z$ is in the single choice of context variable $\tilde{z}_{t-1}$. The full approximate inference algorithm becomes

$$\tilde{P}[\tilde{z}_{t-1}; \alpha_{t-1}] = \alpha_{t-1}/n_y + (1-\alpha_{t-1})\delta_{\tilde{z}_{t-1}}. \quad \mathcal{D}_{t-1} \text{ approximation} \tag{2}$$
$$\tilde{P}[\tilde{z}_t|\tilde{z}_{t-1}; \alpha_{t-1}] = \sum_z \tilde{P}[\tilde{z}_{t-1}=z; \alpha_{t-1}]\mathcal{T}_{zz_t} \quad \text{prior over } z \tag{3}$$
$$\tilde{P}[y_t, \tilde{z}_t|\tilde{z}_{t-1}; \alpha_{t-1}] = \tilde{P}[\tilde{z}_t|\tilde{z}_{t-1}; \alpha_{t-1}]\mathcal{O}_{\tilde{z}_t y_t} \quad \text{propagation to } y \tag{4}$$
$$\tilde{P}[y_t, \tilde{z}_t|\mathcal{D}_t] \propto \tilde{P}[y_t, \tilde{z}_t|\tilde{z}_{t-1}; \alpha_{t-1}]P[\mathbf{x}_t|y_t] \quad \text{conditioning} \tag{5}$$
$$\tilde{P}[y_t|\mathcal{D}_t] = \sum_z \tilde{P}[y_t, \tilde{z}_t=z|\mathcal{D}_t] \quad \textbf{marginalization} \tag{6}$$
$$\tilde{P}[\tilde{z}_t|\mathcal{D}_t] = \sum_y \tilde{P}[y_t=y, \tilde{z}_t|\mathcal{D}_t] \quad \text{marginalization} \tag{7}$$
$$\tilde{z}_t = \text{argmax}_z \tilde{P}[\tilde{z}_t=z|\mathcal{D}_t] \quad \textbf{contextual inference} \tag{8}$$
$$\tilde{\alpha}_t = 1 - \max_z \tilde{P}[\tilde{z}_t=z|\mathcal{D}_t] \quad \textbf{ACh level} \tag{9}$$

where $\mathbf{x}_t, \tilde{z}_{t-1}, \alpha_{t-1}$ are used as approximate sufficient statistics for $\mathcal{D}_t$, the number of $y$ states is $n_y$ (here $n_y = 4$), $\delta_{ij}$ is the Kronecker delta, and the constant of proportionality in equation 5 normalizes the full conditional distribution. The last two lines show the information that is propagated to the next time step; equation 6 shows the representational answer from the model, the distribution over $y_t$ given $\mathcal{D}_t$. These computations are all local and straightforward, except for the representation and normalization of the joint distribution over $y_t$ and $\tilde{z}_t$, a point to which we return later. Crucially, ACh exerts its influence through equation 2. If $\alpha_{t-1}$ is high, then the input stimulus controlled, likelihood term dominates in the conditioning process (equation 5); if $\alpha_{t-1}$ is low, then temporal context ($\tilde{z}_{t-1}$) and the likelihood terms balance.

One potentially dangerous aspect of this inference procedure is that it might get unreasonably committed to a single state $\tilde{z}_{t-1} = \tilde{z}_t = \ldots$ because it does not represent explicitly the probability accorded to the other possible values of $z_{t-1}$ given $\mathcal{D}_{t-1}$. A natural way to avoid this is to bound the ACh level from below by a constant, $\varphi$, making approximate inference slightly more stimulus-bound than exact inference. This approximation should add robustness. In practice, rather than use equation 9, we use

$$\tilde{\alpha}_t = \varphi + (1-\varphi)(1 - \max_z \tilde{P}[\tilde{z}_t=z|\mathcal{D}_t]) \tag{10}$$

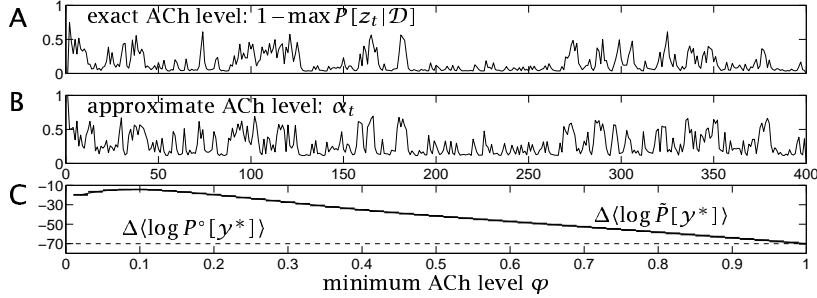

**Figure 4**: ACh model. A) ACh level from the exact posterior for one run. B) ACh level $\alpha_t$ in the approximate model in the same run. Note the coarse similarity between A and B. C) Solid: the mean extra representational cost for the true state $y_t^*$ over that in the exact posterior using the ACh model as a function of the minimum allowed ACh level $\varphi$. Dashed: the same quantity for the pure bottom-up model (which is equivalent to the approximate model for $\varphi = 1$). Errorbars (which are almost invisible) show standard errors of the means over 1000 trials.

Figure 4B shows the approximate ACh level for the same case as figure 4A, using $\varphi = 0.1$. Although the detailed value of this signal is clearly different from that arising from an exact knowledge of the posterior probabilities (in figure 4A), the gross movements are quite similar. Note the effect of $\varphi$ in preventing the ACh level from dropping to 0. Figure 3C shows that the ACh-based approximate posterior values $\tilde{P}[y|\mathcal{D}]$ are much closer to the true values than for the purely bottom-up model, particularly for values of $P[y_t|\mathcal{D}_t]$ near 0 and 1, where most data lie. Figure 3F shows that inference about $z$ is noisy, but the pattern of true values $z_t^*$ is certainly visible. Figure 4C shows the effect of changing $\varphi$ on the quality of inference about the true states $y_t^*$. This shows differences between approximate and exact log probabilities of the true states $y_t^*$, averaged over 1000 cases. If $\varphi = 1$, then inference is completely stimulus-bound, just like the purely bottom-up model; values of $\varphi$ less than 0.2 appear to do well for this and other settings of the parameters of the problem. An upper bound on the performance of approximate inference can be calculated in three steps by: i) using the exact posterior to work out $\tilde{z}_t$ and $\alpha_t$, ii) using these values to approximate $\tilde{P}[\tilde{z}_t; \alpha_t]$ as in equation 2, and iii) using this approximate distribution in equation 4 and the remaining equations. The average resulting cost (*ie* the average resulting difference from the log probability under exact inference) is $-3.5$ log units. Thus, the ACh-based approximation performs well, and much better than purely bottom-up inference.

## 4 Discussion

We have suggested that one of the roles of ACh in cortical processing is to report contextual uncertainty in order to control the balance between stimulus-bound, bottom-up, processing, and contextually-bound, top-down processing. We used the example of a hierarchical HMM in which representational inference for a middle layer should correctly reflect such a balance, and showed that a simple model of the drive and effects of ACh leads to competent inference.

This model is clearly overly simple. In particular, it uses a localist representation for the state $z$, and so exact inference would be feasible. In a more realistic case, distributed representations would be used at multiple levels in the hierarchy, and so only one single context could be entertained at once. Then, it would also not be possible to represent the degree of uncertainty using the level of activities of the

units representing the context, at least given a population-coded representation. It would also be necessary to modify the steps in equations 4 and 5, since it would be hard to represent the joint uncertainty over representations at multiple levels in the hierarchy. Nevertheless, our model shows the feasibility of using an ACh signal in helping propagate and use approximate information over time.

Since it is straightforward to administer cholinergic agonists and antagonists, there are many ways to test aspects of this proposal. We plan to start by using the paradigm of Ress *et al*,[24] which uses fMRI techniques to study bottom-up and top-down influences on the detection of simple visual targets. Preliminary simulation studies indicate that a hidden Markov model under controllable cholinergic modulation can capture several aspects of existent data on animal signal detection tasks.[18]

**Acknowledgements**

We are very grateful to Michael Hasselmo, David Heeger, Sham Kakade and Szabolcs Káli for helpful discussions. Funding was from the Gatsby Charitable Foundation and the NSF. Reference [28] is an extended version of this paper.

# References

[1] Carpenter, GA & Grossberg, S, editors (1991) *Pattern Recognition by Self-Organizing Neural Networks*. Cambridge, MA: MIT Press.

[2] Dayan, P (1999). Recurrent sampling models for the Helmholtz machine. *Neural Computation*, **11**:653-677.

[3] Dayan, P, Hinton, GE, Neal, RM & Zemel, RS (1995) The Helmholtz machine. *Neural Computation* **7**:889-904.

[4] Dayan, P, Kakade, S & Montague, PR (2000). Learning and selective attention. *Nature Neuroscience*, **3**:1218-1223.

[5] Doya, K (1999) Metalearning, neuromodulation and emotion. *The 13th Toyota Conference on Affective Minds*, 46-47.

[6] Everitt, BJ & Robbins, TW (1997) Central cholinergic systems and cognition. *Annual Review of Psychology* **48**:649-684.

[7] Fellous, J-M, Linster, C (1998) Computational models of neuromodulation. *Neural Computation* **10**:771-805.

[8] Grenander, U (1976-1981) *Lectures in Pattern Theory I, II and III: Pattern Analysis, Pattern Synthesis and Regular Structures*. Berlin:Springer-Verlag.

[9] Hasselmo, ME (1995) Neuromodulation and cortical function: Modeling the physiological basis of behavior. *Behavioural Brain Research* **67**:1-27.

[10] Hasselmo, M (1999) Neuromodulation: acetylcholine and memory consolidation. *Trends in Cognitive Sciences* **3**:351-359.

[11] Hasselmo, ME & Bower, JM (1993) Acetylcholine and memory. *Trends in Neurosciences* **16**:218-222.

[12] Hasselmo, ME, Wyble, BP & Wallenstein, GV (1996) Encoding and retrieval of episodic memories: Role of cholinergic and GABAergic modulation in the hippocampus. *Hippocampus* **6**:693-708.

[13] Hinton, GE, & Ghahramani, Z (1997) Generative models for discovering sparse distributed representations. *Philosophical Transactions of the Royal Society of London*. **B352**:1177-1190.

[14] Holland, PC (1997) Brain mechanisms for changes in processing of conditioned stimuli in Pavlovian conditioning: Implications for behavior theory. *Animal Learning & Behavior* **25**:373-399.

[15] Holland, PC & Gallagher, M (1999) Amygdala circuitry in attentional and representational processes. *Trends In Cognitive Sciences* **3**:65-73.

[16] Kakade, S & Dayan, P (2000). Dopamine bonuses. In TK Leen, TG Dietterich & V Tresp, editors, *NIPS 2000.*

[17] MacKay, DM (1956) The epistemological problem for automata. In CE Shannon & J McCarthy, editors, *Automata Studies*. Princeton, NJ: Princeton University Press, 235-251.

[18] McGaughy, J, Kaiser, T, & Sarter, M. (1996). Behavioral vigilance following infusions of 192 IgG=saporin into the basal forebrain: selectivity of the behavioral impairment and relation to cortical AChE-positive fiber density. *Behavioral Neuroscience* **110**: 247-265.

[19] Mumford, D (1994) Neuronal architectures for pattern-theoretic problems. In C Koch & J Davis, editors, *Large-Scale Theories of the Cortex*. Cambridge, MA:MIT Press, 125-152.

[20] Pearce, JM & Hall, G (1980) A model for Pavlovian learning: Variation in the effectiveness of conditioned but not unconditioned stimuli. *Psychological Review* **87**:532-552.

[21] Pfluger, HJ (1999) Neuromodulation during motor development and behavior. *Current Opinion in Neurobiology* **9**:683-689.

[22] Rabiner, LR (1989) A tutorial on hidden Markov models and selected applications in speech recognition. *Proceedings of the IEEE* **77**:257-286.

[23] Rao, RPN & Ballard, DH (1997) Dynamic model of visual recognition predicts neural response properties in the visual cortex. *Neural Computation* **9**:721-763.

[24] Ress, D, Backus, BT & Heeger, DJ (2000) Activity in primary visual cortex predicts performance in a visual detection task. *Nature Neuroscience* **3**:940-945.

[25] Sarter, M, Bruno, JP (1997) Cognitive functions of cortical acetylcholine: Toward a unifying hypothesis. *Brain Research Reviews* **23**:28-46.

[26] Schultz, W (1998) Predictive reward signal of dopamine neurons. *Journal of Neurophysiology* **80**:1–27.

[27] Schultz, W, Dayan, P & Montague, PR (1997). A neural substrate of prediction and reward. *Science,* **275**, 1593-1599.

[28] Yu, A & Dayan, P (2002). Acetylcholine in cortical inference. Submitted to *Neural Networks*.
